# VLSI Implementation of Motion Centroid Localization for Autonomous Navigation

**Ralph Etienne-Cummings**
Dept. of ECE,
Johns Hopkins University,
Baltimore, MD

**Viktor Gruev**
Dept. of ECE,
Johns Hopkins University,
Baltimore, MD

**Mohammed Abdel Ghani**
Dept. of EE,
S. Illinois University,
Carbondale, IL

## Abstract

A circuit for fast, compact and low-power focal-plane motion centroid localization is presented. This chip, which uses mixed signal CMOS components to implement photodetection, edge detection, ON-set detection and centroid localization, models the retina and superior colliculus. The centroid localization circuit uses time-windowed asynchronously triggered row and column address events and two linear resistive grids to provide the analog coordinates of the motion centroid. This VLSI chip is used to realize fast lightweight autonavigating vehicles. The obstacle avoiding line-following algorithm is discussed.

## 1 INTRODUCTION

Many neuromorphic chips which mimic the analog and parallel characteristics of visual, auditory and cortical neural circuits have been designed [Mead, 1989; Koch, 1995]. Recently researchers have started to combine digital circuits with neuromorphic aVLSI systems [Boahen, 1996]. The persistent doctrine, however, has been that computation should be performed in analog, and only communication should use digital circuits. We have argued that hybrid computational systems are better equipped to handle the high speed processing required for real-world problem solving, while maintaining compatibility with the ubiquitous digital computer [Etienne, 1998]. As a further illustration of this point of view, this paper presents a departure form traditional approaches for focal plane centroid localization by offering a mixed signal solution that is simultaneously high-speed, low power and compact. In addition, the chip is interfaced with an 8-bit microcomputer to implement fast autonomous navigation.

Implementation of centroid localization has been either completely analog or completely digital. The analog implementations, realized in the early 1990s, used focal plane current mode circuits to find a global continuos time centroid of the pixels' intensities [DeWeerth, 1992]. Due to their sub-threshold operation, these circuits are low power, but slow. On the other hand, the digital solutions do not compute the centroid at the focal

plane. They use standard CCD cameras, A/D converters and DSP/CPU to compute the intensity centroid [Mansfield, 1996]. These software approaches offer multiple centroid localization with complex mathematical processing. However, they suffer from the usual high power consumption and non-scalability of traditional digital visual processing systems. Our approach is novel in many aspects. We benefit from the low power, compactness and parallel organization of focal plane analog circuits and the speed, robustness and standard architecture of asynchronous digital circuits. Furthermore, it uses event triggered analog address read-out, which is ideal for the visual centroid localization problem. Moreover, our chip responds to moving targets only by using the ON-set of each pixel in the centroid computation. Lastly, our chip models the retina and two dimensional saccade motor error maps of superior colliculus on a single chip [Sparks, 1990]. Subsequently, this chip is interfaced with a µC for autonomous obstacle avoidance during line-following navigation. The line-following task is similar to target tracking using the saccadic system, except that the "eye" is fixed and the "head" (the vehicle) moves to maintain fixation on the target. Control signals provided to the vehicle based on decisions made by the µC are used for steering and accelerating/braking. Here the computational flexibility and programmability of the µC allows rapid prototyping of complex and robust algorithms.

## 2 CENTROID LOCALIZATION

The mathematical computation of the centroid of an object on the focal plane uses intensity weighted average of the position of the pixels forming the object [DeWeerth, 1992]. Equation (1) shows this formulation. The implementation of this representation

$$\hat{x} = \frac{\sum_{i=1}^{N} I_i x_i}{\sum_{i=1}^{N} I_i} \quad and \quad \hat{y} = \frac{\sum_{i=1}^{N} I_i y_i}{\sum_{i=1}^{N} I_i} \quad (1)$$

can be quite involved since a product between the intensity and position is implied. To eliminate this requirement, the intensity of the pixels can be normalized to a single value within the object. This gives equation (2) since the intensity can be factored out of the summations. Normalization of the intensity using a simple threshold is not advised since

$$\hat{x} = \frac{\sum_{i=1}^{N} x_i}{N} \quad and \quad \hat{y} = \frac{\sum_{i=1}^{N} y_i}{N} \quad (2)$$

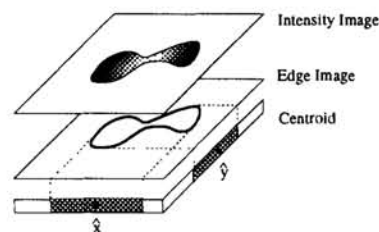

Figure 1: Centroid computation architecture.

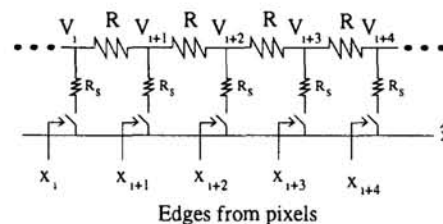

Figure 2: Centroid computation method.

the value of the threshold is dependent on the brightness of the image and number of pixels forming the object may be altered by the thresholding process. To circumvent these problems, we take the view that the centroid of the object is defined in relation to its boundaries. This implies that edge detection (second order spatial derivative of intensity) can be used to highlight the boundaries, and edge labeling (the zero-crossing of the edges) can be used to normalize the magnitude of the edges. Subsequently, the centroid

of the zero-crossings is computed. Equation (2) is then realized by projecting the zero-crossing image onto the x- and y-axis and performing two linear centroid determinations. Figure (1) shows this process.

The determination of the centroid is computed using a resistance grid to associate the position of a column (row) with a voltage. In figure 2, the positions are given by the voltages $V_i$. By activating the column (row) switch when a pixel of the edge image appears in that column (row), the position voltage is connected to the output line through the switch impedance, $R_s$. As more switches are activated, the voltage on the output line approximates equation (2). Clearly, since no buffers are used to isolate the position voltages, as more switches are activated, the position voltages will also change. This does not pose a problem since the switch resistors are design to be larger than the position resistors (the switch currents are small compared to the grid current). Equation (3) gives the error between the ideal centroid and the switch loaded centroid in the worst case when $R_s = 0\Omega$. In the equation, N is the number of nodes, M is the number of switches set and $x_1$ and $x_M$ are the locations of the first and last set switches, respectively. This error is improved as $R_s$ gets larger, and vanishes as N (M$\leq$N) approaches infinity. The terms $x_i$ represent an ascending ordered list of the activated switches; $x_1$ may correspond to column five, for example. This circuit is compact since it uses only a simple linear resistive grid and MOS switches. It is low power because the total grid resistance, N x R, can be large. It can be fast when the parasitic capacitors are kept small. It provides an analog position value, but it is triggered by fast digital signals that activate the switches.

$$error = \frac{V_{max} - V_{min}}{M(N+1)} \sum_{i=1}^{M} \left[ x_i - \frac{x_1(N+1)}{N+1+x_1-x_m} \right] \qquad (3)$$

# 3 MODELING THE RETINA AND SUPERIOR COLLICULUS

## 3.1 System Overview

The centroid computation approach presented in section 2 is used to isolate the location of moving targets on a 2D focal plane array. Consequently, a chip which realizes a neuromorphic visual target acquisition system based on the saccadic generation mechanism of primates can be implemented. The biological saccade generation process is mediated by the superior colliculus, which contains a map of the visual field [Sparks, 1990]. In laboratory experiments, cellular recordings suggest that the superior colliculus provides the spatial location of targets to be foveated. Clearly, a great deal of neural circuitry exists between the superior colliculus and the eye muscle. Horiuchi has built an analog system which replicates most of the neural circuits (including the motor system) which are believed to form the saccadic system [Horiuchi, 1996]. Staying true to the anatomy forced his implementation to be a complex multi-chip system with many control parameters. On the other hand, our approach focuses on realizing a compact single chip solution by only mimicking the behavior of the saccadic system, but not its structure.

## 3.2 Hardware Implementation

Our approach uses a combination of analog and digital circuits to implement the functions of the retina and superior colliculus at the focal plane. We use simple digital control ideas, such as pulse-width modulation and stepper motors, to position the "eye". The retina portion of this chip uses photodiodes, logarithmic compression, edge detection and zero-crossing circuits. These circuits mimic the first three layers of cells in the retina

with mixed sub-threshold and strong inversion circuits. The edge detection circuit is realized with an approximation of the Laplacian operator implemented using the difference between a smooth (with a resistive grid) and unsmoothed version of the image [Mead, 1989]. The high gain of the difference circuit creates a binary image of approximate zero-crossings. After this point, the computation is performed using mixed analog/digital circuits. The zero-crossings are fed to ON-set detectors (positive temporal derivatives) which signal the location of moving or flashing targets. These circuits model the behavior of some of the amacrine and ganglion cells of the primate retina [Barlow, 1982]. These first layers of processing constitute all the "direct" mimicry of the biological models. Figure 3 shows the schematic of these early processing layers.

The ON-set detectors provide inputs to the model of the superior colliculus circuits. The ON-set detectors allow us to segment moving targets against textured backgrounds. This is an improvement on earlier centroid and saccade chips that used pixel intensity. The essence of the superior colliculus map is to locate the target that is to be foveated. In our case, the target chosen to be foveated will be moving. Here motion is define simply as the change in contrast over time. Motion, in this sense, can be seen as being the earliest measurable attribute of the target which can trigger a saccade without requiring any high level decision making. Subsequently, the coordinates of the motion must be extracted and provided to the motor drivers.

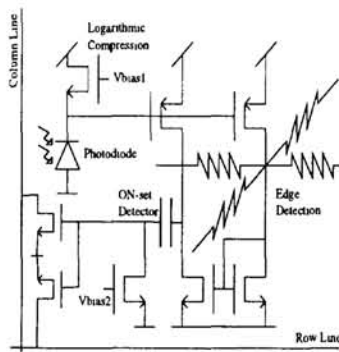

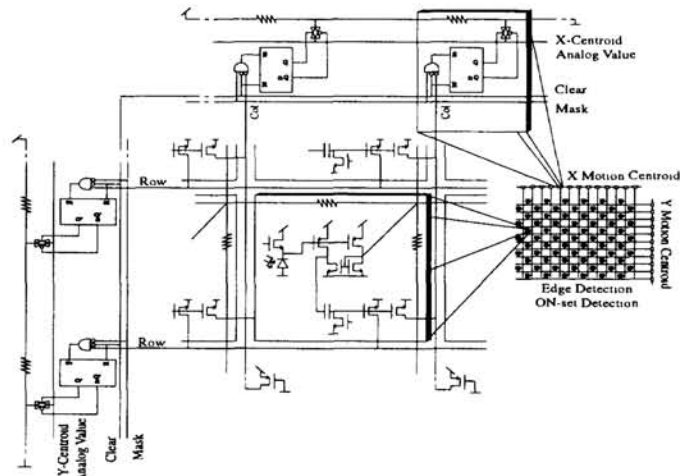

Figure 3: Schematic of the model of the retina.

Figure 4: Schematic of the model of the superior colliculus.

The circuits for locating the target are implemented entirely with mixed signal, non-neuromorphic circuits. The theoretical foundation for our approach is presented in section 2. The ON-set detector is triggered when an edge of the target appears at a pixel. At this time, the pixel broadcasts its location to the edge of the array by activating row and column lines. This row (column) signal sets a latch at the right (top) of the array. The latches asynchronously activate switches and the centroid of the activated positions is provided. The latches remain set until they are cleared by an external control signal. This control signal provides a time-window over which the centroid output is integrated. This has the effect of reducing noise by combining the outputs of pixels which are activated at different instances even if they are triggered by the same motion (an artifact of small fill factor focal plane image processing). Furthermore, the latches can be masked from the pixels' output with a second control signal. This signal is used to de-activate the centroid

circuit during a saccade (saccadic suppression). A centroid valid signal is also generated by the chip. Figure 4 shows a portion of the schematic of the superior colliculus model.

## 3.3 Results

In contrast to previous work, this chip provides the 2-D coordinates of the centroid of a moving target. Figure 5 shows the oscilloscope trace of the coordinates as a target moves back and forth, in and out of the chip's field of view. The y-coordinate does change while the x-coordinate increases and decreases as the target moves to the left and right, respectively. The chip has been used to track targets in 2-D by making micro-saccades. In this case, the chip chases the target as it attempts to escape from the center. The eye movement is performed by converting the analog coordinates into PWM signals, which are used to drive stepper motors. The system performance is limited by the contrast sensitivity of the edge detection circuit, and the frequency response of the edge (high frequency cut-off) and ON-set (low frequency cut-off) detectors. With the appropriate optics, it can track walking or running persons under indoor or outdoor lighting conditions at close or far distances. Table I gives a summary of the chip characteristics.

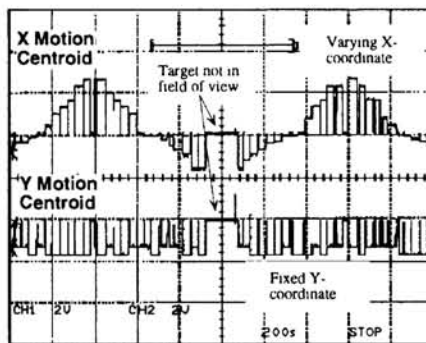

Figure 5: Oscilloscope trace of 2D centroid for a moving target.

| Technology | 1.2um ORBIT |
|---|---|
| Chip Size | 4 mm$^2$ |
| Array Size | 12 x 10 |
| Pixel Size | 110 x 110 um |
| Fill Factor | 11% |
| Intensity | 0.1u - 100m W/cm$^2$ |
| Min. Contrast | 10% |
| Response Time | 2 - 10$^6$ Hz (@1 m W/cm$^2$) |
| Power (chip) | 5 mW (@1 m W/cm$^2$, Vdd = 6V) |

Table I: Chip characteristics.

# 4 APPLICATION: OBSTACLE AVOIDANCE DURING LINE-FOLLOWING AUTONAVIGATION

## 4.1 System Overview

The frenzy of activity towards developing neuromorphic systems over the pass 10 years has been mainly driven by the promise that one day engineers will develop machines that can interact with the environment in a similar way as biological organisms. The prospect of having a robot that can help humans in their daily tasks has been a dream of science fiction for many decades. As can be expected, the key to success is premised on the development of compact systems, with large computational capabilities, at low cost (in terms of hardware and power). Neuromorphic VLSI systems have closed the gap between dreams and reality, but we are still very far from the android robot. For all these robots, autonomous behavior, in the form of auto-navigation in natural environments, must be one of their primary skills. For miniaturization, neuromorphic vision systems performing most of the pre-processing, can be coupled with small fast computers to realize these compact yet powerful sensor/processor modules.

## 4.2 Navigation Algorithm

The simplest form of data driven auto-navigation is the line-following task. In this task, the robot must maintain a certain relationship with some visual cues that guide its motion. In the case of the line-follower, the visual system provides data regarding the state of the

line relative to the vehicle, which results in controlling steering and/or speed. If obstacle avoidance is also required, auto-navigation is considerably more difficult. Our system handles line-following and obstacle avoidance by using two neuromorphic visual sensors that provide information to a micro-controller (μC) to steer, accelerate or decelerate the vehicle. The sensors, which uses the centroid location system outlined above, provides information on the position of the line and obstacles to the μC, which provides PWM signals to the servos for controlling the vehicle. The algorithm implemented in the μC places the two sensors in competition with each other to force the line into a blind zone between the sensors. Simultaneously, if an object enters the visual field from outside, it is treated as an obstacle and the μC turns the car away from the object. Obstacle avoidance is given higher priority than line-following to avoid collisions. The μC also keeps track of the direction of avoidance such that the vehicle can be re-oriented towards the line after the obstacle is pushed out of the field of view. Lastly, for line following, the position, orientation and velocity of drift, determined from the temporal derivative of the centroid, are used to track the line. The control strategy is to keep the line in the blind zone, while slowing down at corners, speeding up on straight aways and avoiding obstacles. The angle which the line or obstacle form with the x-axis also affects the speed. The value of the x-centroid relative to the y-centroid provides rudimentary estimate of the orientation of the line or obstacle to the vehicle. For example, angles less

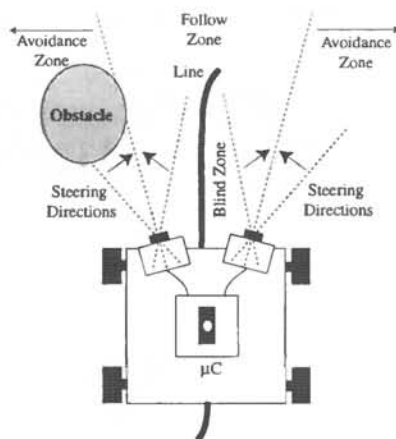

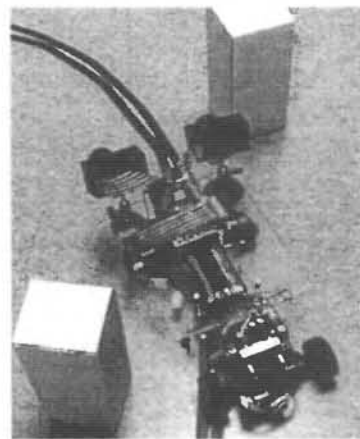

Figure 6: Block diagram of the autonomous line-follower system.

Figure 7: A picture of the vehicle.

(greater) than +/- 45 degrees tend to have small (large) x-coordinates and large (small) y-coordinates and require deceleration (acceleration). Figure 6 shows the organization of the sensors on the vehicle and control spatial zones. Figure 7 shows the vehicle and samples of the line and obstacles.

## 4.3 Hardware Implementation

The coordinates from the centroid localization circuits are presented to the μC for analysis. The μC used is the Microchip PIC16C74. This chip is chosen because of its five A/D inputs and three PWM outputs. The analog coordinates are presented directly to the A/D inputs. Two of the PWM outputs are connected to the steering and speed control servos. The PIC16C74 runs at 20 MHz and has 35 instructions, 4K by 8-b ROM and 80 by 20-b RAM. The program which runs on the PIC determines the control action to take, based on the signal provided by the neuromorphic visual sensors. The vehicle used is a four-wheel drive radio controlled model car (the radio receiver is disconnected) with Digital Proportional Steering (DPS).

## 4.4 Results

The vehicle was tested on a track composed of black tape on a gray linoleum floor with black and white obstacles. The track formed a closed loop with two sharp turns and some smooth S-curves. The neuromorphic vision chip was equipped with a 12.5 mm variable iris lens, which limited its field of view to about 10°. Despite the narrow field of view, the car was able to navigate the track at an average speed of 1 m/s without making any errors. On less curvy parts of the track, it accelerated to about 2 m/s and slowed down at the corners. When the speed of the vehicle is scaled up, the errors made are mainly due to over steering.

## 5  CONCLUSION

A 2D model of the saccade generating components of the superior colliculus is presented. This model only mimics the functionality the saccadic system using mixed signal focal plane circuits that realize motion centroid localization. The single chip combines a silicon retina with the superior colliculus model using compact, low power and fast circuits. Finally, the centroid chip is interfaced with an 8-bit μC and vehicle for fast line-following autonavigation with obstacle avoidance. Here all of the required computation is performed at the visual sensor, and a standard μC is the high-level decision maker.

### References

Barlow H., *The Senses: Physiology of the Retina*, Cambridge University Press, Cambridge, England, 1982.

Boahen K., "Retinomorphic Vision Systems II: Communication Channel Design," *ISCAS 96*, Atlanta, GA, 1996.

DeWeerth, S. P., "Analog VLSI Circuits for Stimulus Localization and Centroid Computation," *Int'l J. Computer Vision*, Vol. 8, No. 2, pp. 191-202, 1992.

Etienne-Cummings R., J Van der Spiegel and P. Mueller, "Neuromorphic and Digital Hybrid Systems," *Neuromorphic Systems: Engineering Silicon from Neurobiology*, L. Smith and A. Hamilton (Eds.), World Scientific, 1998.

Horiuchi T., T. Morris, C. Koch and S. P. DeWeerth, "Analog VLSI Circuits for Attention-Based Visual Tracking," *Advances in Neural Information Processing Systems*, Vol. 9, Denver, CO, 1996.

Koch C. and H. Li (Eds.), *Vision Chips: Implementing Vision Algorithms with Analog VLSI Circuits*, IEEE Computer Press, 1995.

Mansfield, P., "Machine Vision Tackles Star Tracking," *Laser Focus World*, Vol. 30, No. 26, pp. S21-S24, 1996.

Mead C. and M. Ismail (Eds.), *Analog VLSI Implementation of Neural Networks*, Kluwer Academic Press, Newell, MA, 1989.

Sparks D., C. Lee and W. Rohrer, "Population Coding of the Direction, Amplitude and Velocity of Saccadic Eye Movements by Neurons in the Superior Colliculus," *Proc. Cold Spring Harbor Symp. Quantitative Biology*, Vol. LV, 1990.
